# A Growing Neural Gas Network Learns Topologies

**Bernd Fritzke**
Institut für Neuroinformatik
Ruhr-Universität Bochum
D-44780 Bochum
Germany

## Abstract

An incremental network model is introduced which is able to learn the important topological relations in a given set of input vectors by means of a simple Hebb-like learning rule. In contrast to previous approaches like the "neural gas" method of Martinetz and Schulten (1991, 1994), this model has no parameters which change over time and is able to continue learning, adding units and connections, until a performance criterion has been met. Applications of the model include vector quantization, clustering, and interpolation.

## 1 INTRODUCTION

In unsupervised learning settings only input data is available but no information on the desired output. What can the goal of learning be in this situation?

One possible objective is *dimensionality reduction:* finding a low-dimensional subspace of the input vector space containing most or all of the input data. Linear subspaces with this property can be computed directly by principal component analysis or iteratively with a number of network models (Sanger, 1989; Oja, 1982). The Kohonen feature map (Kohonen, 1982) and the "growing cell structures" (Fritzke, 1994b) allow projection onto non-linear, discretely sampled subspaces of a dimensionality which has to be chosen *a priori*. Depending on the relation between inherent data dimensionality and dimensionality of the target space, some information on the topological arrangement of the input data may be lost in the process.

This is not astonishing since a reversible mapping from high-dimensional data to lower-dimensional spaces (or structures) does not exist in general.

Asking how structures must look like to allow reversible mappings directly leads to another possible objective of unsupervised learning which can be described as *topology learning*: Given some high-dimensional data distribution $P(\xi)$, find a topological structure which closely reflects the topology of the data distribution. An elegant method to construct such structures is "competitive Hebbian learning" (CHL) (Martinetz, 1993). CHL requires the use of some vector quantization method. Martinetz and Schulten propose the "neural gas" (NG) method for this purpose (Martinetz and Schulten, 1991).

We will briefly introduce and discuss the approach of Martinetz and Schulten. Then we propose a new network model which also makes use of CHL. In contrast to the above-mentioned CHL/NG combination, this model is incremental and has only constant parameters. This leads to a number of advantages over the previous approach.

## 2 COMPETITIVE HEBBIAN LEARNING AND NEURAL GAS

CHL (Martinetz, 1993) assumes a number of centers in $\mathbf{R}^n$ and successively inserts topological connections among them by evaluating input signals drawn from a data distribution $P(\xi)$. The principle of this method is:

> For each input signal $x$ connect the two closest centers (measured by Euclidean distance) by an edge.

The resulting graph is a subgraph of the Delaunay triangulation (fig. 1a) corresponding to the set of centers. This subgraph (fig. 1b), which is called the "induced Delaunay triangulation", is limited to those areas of the input space $\mathbf{R}^n$ where $P(\xi) > 0$. The "induced Delaunay triangulation" has been shown to optimally preserve topology in a very general sense (Martinetz, 1993).

Only centers lying on the input data submanifold or in its vicinity actually develop any edges. The others are useless for the purpose of topology learning and are often called *dead units*. To make use of all centers they have to be placed in those regions of $\mathbf{R}^n$ where $P(\xi)$ differs from zero. This could be done by any vector quantization (VQ) procedure. Martinetz and Schulten have proposed a particular kind of VQ method, the mentioned NG method (Martinetz and Schulten, 1991). The main principle of NG is the following:

> For each input signal $x$ adapt the $k$ nearest centers whereby $k$ is decreasing from a large initial to a small final value.

A large initial value of $k$ causes adaptation (movement towards the input signal) of a large fraction of the centers. Then $k$ (the adaptation range) is decreased until finally only the nearest center for each input signal is adapted. The adaptation strength underlies a similar decay schedule. To realize the parameter decay one has to define the total number of adaptation steps for the NG method in advance.

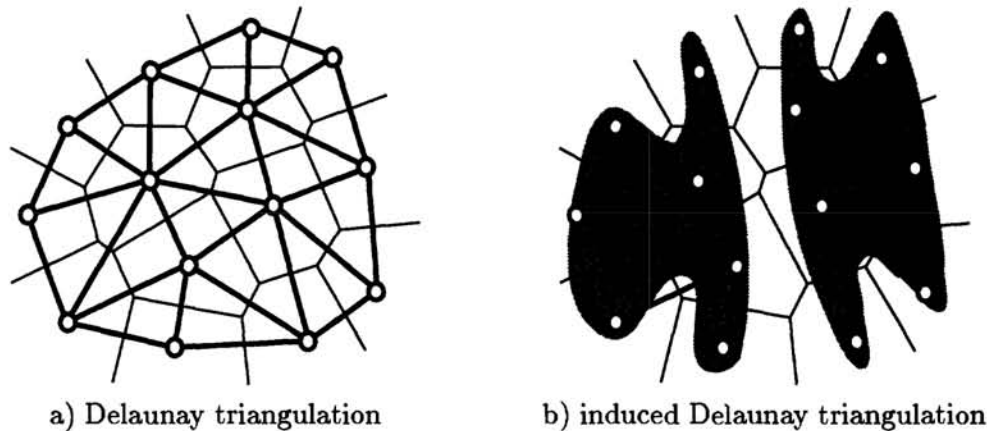

a) Delaunay triangulation        b) induced Delaunay triangulation

Figure 1: Two ways of defining closeness among a set of points. a) The *Delaunay triangulation* (thick lines) connects points having neighboring *Voronoi polygons* (thin lines). Basically this reduces to points having small Euclidean distance w.r.t. the given set of points. b) The *induced Delaunay triangulation* (thick lines) is obtained by masking the original Delaunay triangulation with a data distribution $P(\xi)$ (shaded). Two centers are only connected if the common border of their Voronoi polygons lies at least partially in a region where $P(\xi) > 0$ (closely adapted from Martinetz and Schulten, 1994)

For a given data distribution one could now first run the NG algorithm to distribute a certain number of centers and then use CHL to generate the topology. It is, however, also possible to apply both techniques concurrently (Martinetz and Schulten, 1991). In this case a method for removing obsolete edges is required since the motion of the centers may make edges invalid which have been generated earlier. Martinetz and Schulten use an *edge aging* scheme for this purpose. One should note that the CHL algorithm does not influence the outcome of the NG method in any way since the adaptations in NG are based only on distance in input space and not on the network topology. On the other hand NG does influence the topology generated by CHL since it moves the centers around.

The combination of NG and CHL described above is an effective method for topology learning. A problem in practical applications, however, may be to determine *a priori* a suitable number of centers. Depending on the complexity of the data distribution which one wants to model, very different numbers of centers may be appropriate. The nature of the NG algorithm requires a decision in advance and, if the result is not satisfying, one or several new simulations have to be performed from scratch. In the following we propose a method which overcomes this problem and offers a number of other advantages through a flexible scheme for center insertion.

## 3   THE GROWING NEURAL GAS ALGORITHM

In the following we consider networks consisting of

- a set $A$ of units (or nodes). Each unit $c \in A$ has an associated *reference vector* $w_c \in \mathbf{R}^n$. The reference vectors can be regarded as positions in input space of the corresponding units.
- a set $N$ of connections (or edges) among pairs of units. These connections are not weighted. Their sole purpose is the definition of topological structure.

Moreover, there is a (possibly infinite) number of $n$-dimensional input signals obeying some unknown probability density function $P(\xi)$.

The main idea of the method is to successively add new units to an initially small network by evaluating local statistical measures gathered during previous adaptation steps. This is the same approach as used in the "growing cell structures" model (Fritzke, 1994b) which, however, has a topology with a fixed dimensionality (e.g., two or three).

In the approach described here, the network topology is generated incrementally by CHL and has a dimensionality which depends on the input data and may vary locally. The complete algorithm for our model which we call "growing neural gas" is given by the following:

0. Start with two units $a$ and $b$ at random positions $w_a$ and $w_b$ in $\mathbf{R}^n$.

1. Generate an input signal $\xi$ according to $P(\xi)$.

2. Find the nearest unit $s_1$ and the second-nearest unit $s_2$.

3. Increment the age of all edges emanating from $s_1$.

4. Add the squared distance between the input signal and the nearest unit in input space to a local counter variable:

$$\Delta\text{error}(s_1) = \|w_{s_1} - \xi\|^2$$

5. Move $s_1$ and its direct topological neighbors[1] towards $\xi$ by fractions $\epsilon_b$ and $\epsilon_n$, respectively, of the total distance:

$$\begin{aligned} \Delta w_{s_1} &= \epsilon_b(\xi - w_{s_1}) \\ \Delta w_n &= \epsilon_n(\xi - w_n) \text{ for all direct neighbors } n \text{ of } s_1 \end{aligned}$$

6. If $s_1$ and $s_2$ are connected by an edge, set the age of this edge to zero. If such an edge does not exist, create it.[2]

7. Remove edges with an age larger than $a_{max}$. If this results in points having no emanating edges, remove them as well.

8. If the number of input signals generated so far is an integer multiple of a parameter $\lambda$, insert a new unit as follows:

   - Determine the unit $q$ with the maximum accumulated error.
   - Insert a new unit $r$ halfway between $q$ and its neighbor $f$ with the largest error variable:

$$w_r = 0.5 \, (w_q + w_f).$$

   - Insert edges connecting the new unit $r$ with units $q$ and $f$, and remove the original edge between $q$ and $f$.
   - Decrease the error variables of $q$ and $f$ by multiplying them with a constant $\alpha$. Initialize the error variable of $r$ with the new value of the error variable of $q$.

9. Decrease all error variables by multiplying them with a constant $d$.

10. If a stopping criterion (e.g., net size or some performance measure) is not yet fulfilled go to step 1.

How does the described method work? The adaptation steps towards the input signals (5.) lead to a general movement of all units towards those areas of the input space where signals come from ($P(\xi) > 0$). The insertion of edges (6.) between the nearest and the second-nearest unit with respect to an input signal generates a single connection of the "induced Delaunay triangulation" (see fig. 1b) *with respect to the current position of all units.*

The removal of edges (7.) is necessary to get rid of those edges which are no longer part of the "induced Delaunay triangulation" because their ending points have moved. This is achieved by *local* edge aging (3.) around the nearest unit combined with age re-setting of those edges (6.) which already exist between **nearest** and second-nearest units.

With insertion and removal of edges the model tries to construct and then track the "induced Delaunay triangulation" which is a slowly moving target due to the adaptation of the reference vectors.

The accumulation of squared distances (4.) during the adaptation helps to identify units lying in areas of the input space where the mapping from signals to units causes much error. To reduce this error, new units are inserted in such regions.

## 4   SIMULATION RESULTS

We will now give some simulation results to demonstrate the general behavior of our model. The probability distribution in fig. 2 has been proposed by Martinetz and Schulten (1991) to demonstrate the non-incremental "neural gas" model. It can be seen that our model quickly learns the important topological relations in this rather complicated distribution by forming structures of different dimensionalities.

The second example (fig. 3) illustrates the differences between the proposed model and the original NG network. Although the final topology is rather similar for both models, intermediate stages are quite different. Both models are able to identify the clusters in the given distribution. Only the "growing neural gas" model, however,

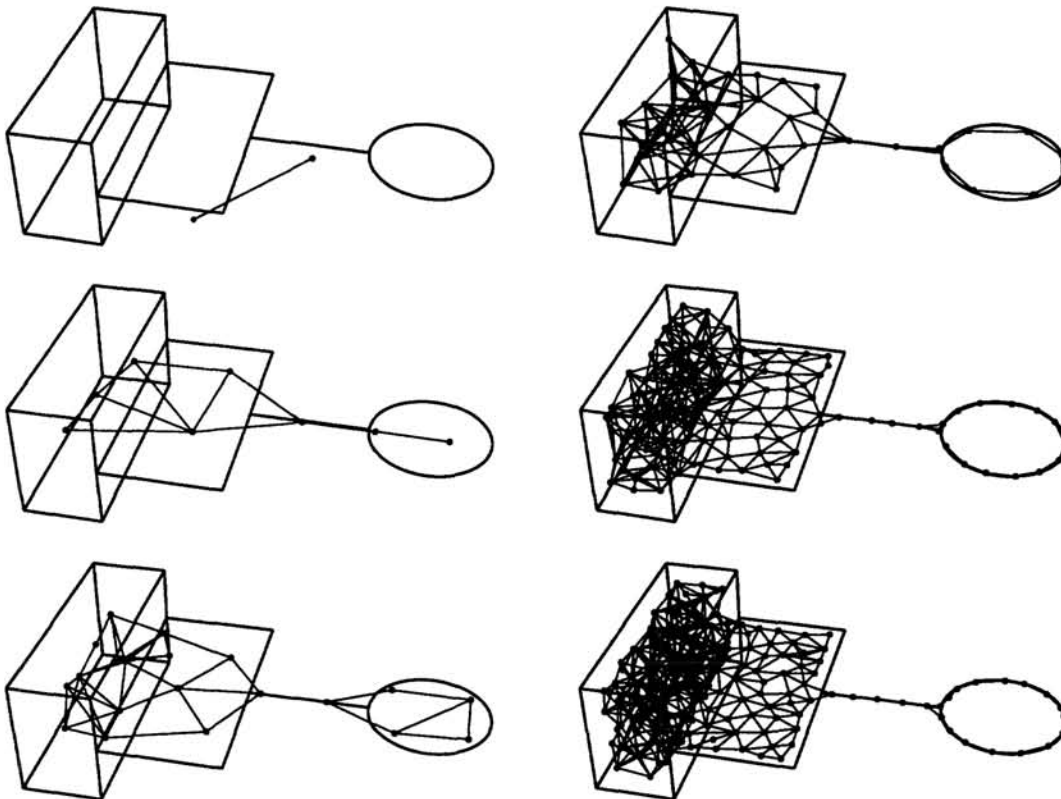

Figure 2: The "growing neural gas" network adapts to a signal distribution which has different dimensionalities in different areas of the input space. Shown are the initial network consisting of two randomly placed units and the networks after 600, 1800, 5000, 15000 and 20000 input signals have been applied. The last network shown is not the necessarily the "final" one since the growth process could in principle be continued indefinitely. The parameters for this simulation were: $\lambda = 100$, $\epsilon_b = 0.2$, $\epsilon_n = 0.006$, $\alpha = 0.5$, $a_{max} = 50$, $d = 0.995$.

could continue to grow to discover still smaller clusters (which are not present in this particular example, though).

## 5  DISCUSSION

The "growing neural gas" network presented here is able to make explicit the important topological relations in a given distribution $P(\xi)$ of input signals. An advantage over the NG method of Martinetz and Schulten is the incremental character of the model which eliminates the need to pre-specify a network size. Instead, the growth process can be continued until a user-defined performance criterion or network size is met. All parameters are constant over time in contrast to other models which heavily rely on decaying parameters (such as the NG method or the Kohonen feature map).

It should be noted that the topology generated by CHL is not an optional feature

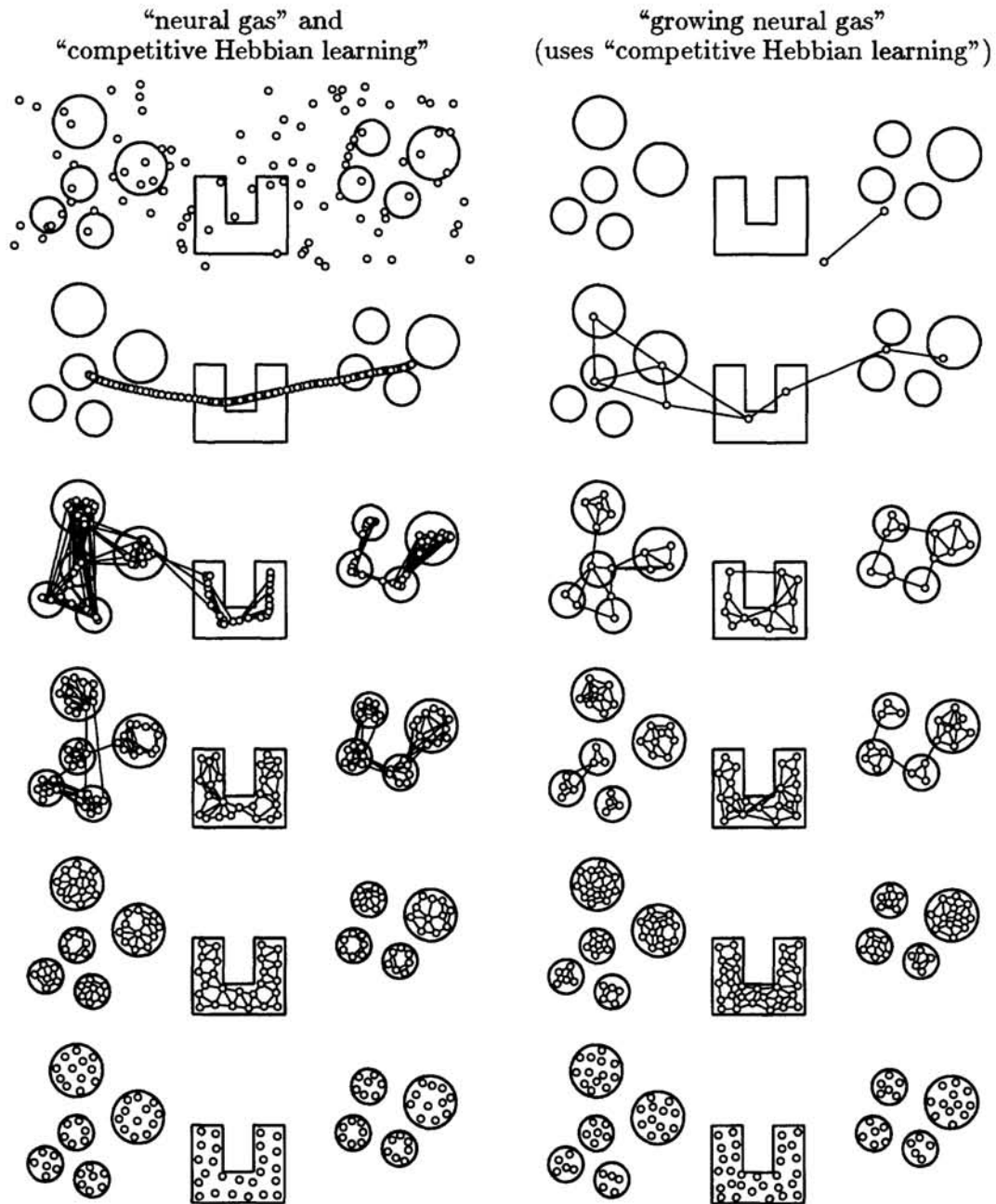

Figure 3: The NG/CHL network of Martinetz and Schulten (1991) and the author's "growing neural gas" model adapt to a clustered probability distribution. Shown are the respective initial states (top row) and a number of intermediate stages. Both the number of units in the NG model and the *final* number of units in the "growing neural gas" model are 100. The bottom row shows the distribution of centers after 10000 adaptation steps (the edges are as in the previous row but not shown). The center distribution is rather similar for both models although the intermediate stages differ significantly.

of our method (as it is for the NG model) but an essential component since it is used to direct the (completely local) adaptation as well as insertion of centers. It is probably the proper initialization of new units by interpolation from existing ones which makes it possible to have only constant parameters and local adaptations.

Possible applications of our model are clustering (as shown) and vector quantization. The network should perform particularly well in situations where the neighborhood information (in the edges) is used to implement interpolation schemes between neighboring units. By using the error occuring in early phases it can be determined where to insert new units to generate a topological look-up table of different density and different dimensionality in particular areas of the input data space.

Another promising direction of research is the combination with supervised learning. This has been done earlier with the "growing cell structures" (Fritzke, 1994c) and recently also with the "growing neural gas" described in this paper (Fritzke, 1994a). A crucial property for this kind of application is the possibility to choose an arbitrary insertion criterion. This is a feature not present, e.g., in the original "growing neural gas". The first results of this new supervised network model, an incremental radial basis function network, are very promising and we are further investigating this currently.

## Footnotes

[1]Throughout this paper the term *neighbors* denotes units which are topological neighbors in the graph (as opposed to units within a small Euclidean distance of each other in input space).

[2]This step is Hebbian in its spirit since correlated activity is used to decide upon insertions.

# References

Fritzke, B. (1994a). Fast learning with incremental rbf networks. *Neural Processing Letters*, 1(1):2–5.

Fritzke, B. (1994b). Growing cell structures – a self-organizing network for unsupervised and supervised learning. *Neural Networks*, 7(9):1441–1460.

Fritzke, B. (1994c). Supervised learning with growing cell structures. In Cowan, J., Tesauro, G., and Alspector, J., editors, *Advances in Neural Information Processing Systems 6*, pages 255–262. Morgan Kaufmann Publishers, San Mateo, CA.

Kohonen, T. (1982). Self-organized formation of topologically correct feature maps. *Biological Cybernetics*, 43:59–69.

Martinetz, T. M. (1993). Competitive Hebbian learning rule forms perfectly topology preserving maps. In *ICANN'93: International Conference on Artificial Neural Networks*, pages 427–434, Amsterdam. Springer.

Martinetz, T. M. and Schulten, K. J. (1991). A "neural-gas" network learns topologies. In Kohonen, T., Mäkisara, K., Simula, O., and Kangas, J., editors, *Artificial Neural Networks*, pages 397–402. North-Holland, Amsterdam.

Martinetz, T. M. and Schulten, K. J. (1994). Topology representing networks. *Neural Networks*, 7(3):507–522.

Oja, E. (1982). A simplified neuron model as a principal component analyzer. *Journal of Mathematical Biology*, 15:267–273.

Sanger, T. D. (1989). An optimality principle for unsupervised learning. In Touretzky, D. S., editor, *Advances in Neural Information Processing Systems 1*, pages 11–19. Morgan Kaufmann, San Mateo, CA.
